# SpikeAnts, a spiking neuron network modelling the emergence of organization in a complex system

**Sylvain Chevallier**
TAO, INRIA-Saclay
Univ. Paris-Sud
F-91405 Orsay, France
sylchev@lri.fr

**Hélène Paugam-Moisy**
LIRIS, CNRS
Univ. Lyon 2
F-69676 Bron, France
hpaugam@liris.cnrs.fr

**Michèle Sebag**
TAO, LRI − CNRS
Univ. Paris-Sud
F-91405 Orsay, France
sebag@lri.fr

## Abstract

Many complex systems, ranging from neural cell assemblies to insect societies, involve and rely on some division of labor. How to enforce such a division in a decentralized and distributed way, is tackled in this paper, using a spiking neuron network architecture. Specifically, a spatio-temporal model called SpikeAnts is shown to enforce the emergence of synchronized activities in an ant colony. Each ant is modelled from two spiking neurons; the ant colony is a sparsely connected spiking neuron network. Each ant makes its decision (among foraging, sleeping and self-grooming) from the competition between its two neurons, after the signals received from its neighbor ants. Interestingly, three types of temporal patterns emerge in the ant colony: asynchronous, synchronous, and synchronous periodic foraging activities − similar to the actual behavior of some living ant colonies. A phase diagram of the emergent activity patterns with respect to two control parameters, respectively accounting for ant sociability and receptivity, is presented and discussed.

## 1   Introduction

The emergence of organization is at the core of many complex systems, from neural cell assemblies to living insect societies. For instance, the emergence of synchronized rhythmical activity has been observed in many social insect colonies [2, 4, 5, 7], where synchronized patterns of activity may indeed contribute to the collective efficiency in various ways. But how do ants proceed to temporally synchronize their activity? As suggested by Cole [4], the synchronization of activity is a consequence of temporal coupling between individuals. It thus comes naturally to investigate how spiking neuron networks (SNNs), also based on temporal dynamics, enable to model the emergence of collective phenomena, specifically synchronized activities, in complex systems. The reader's familiarity with SNNs, inspired from the mechanisms of information processing in the brain, is assumed in the following, referring to [18] for a comprehensive presentation.

### 1.1   Related work

In computational neuroscience, SNNs are well known for generating a rich variety of dynamical patterns of activity, e.g. synchrony of cell assemblies [9], complete synchrony [17], transient synchrony [10], order-chaos phase transition [20] or polychronization [11]. For instance, a mesoscopic model [3] explains the emergence of a rhythmic oscillation at the network level, resulting from the competition of excitatory and inhibitory connections between neurons. In computer science, the field of reservoir computing (RC) [13] focuses on analyzing and exploiting the echos generated by external inputs in the dynamics of sparse random networks. The proposed SpikeAnts model features one distinctive characteristics compared to the state of the art in RC and SNNs: its only aim is to

model an emergent property in a complex closed system; it does neither receive any external inputs nor involve any learning rule. To our best knowledge, current models of emergence are mostly based on statistical physics, involving differential equations and mean field approaches [19], or mathematics and computer science, using random Markov fields, cellular automata or multi-agent systems.

## 1.2 Target of the SpikeAnts model

The SpikeAnts model implements a distributed decision making process in a population of agents, say an ant colony. The phenomenon to analyze is the division of labor. The model relies on the spatio-temporal interactions of spiking neurons, where each ant agent is accounted for by two neurons.

A simplified scheme is proposed, inspired from [2] and [16]: Each agent may be in one out of four states, $\mathcal{O}$bserving, $\mathcal{F}$oraging, $\mathcal{S}$leeping or self-$\mathcal{G}$rooming (Fig. 1). The interactions take place during the observation round. Each agent $a$ observes its environment and if it perceives none or too few working agents, $a$ goes foraging for a given time and eventually goes to sleep. Otherwise, if $a$ perceives "sufficiently many" agents engaged in foraging, it goes back to the nest for less vital tasks (the *grooming* state) before returning to observation after a while. Each state lasts for a fixed duration (resp. $t_{\mathcal{O}}$, $t_{\mathcal{F}}$, $t_{\mathcal{S}}$ and $t_{\mathcal{G}}$), with an exception for the observation state. The observation period is only subject to an upper bound $t_{\mathcal{O}}$. If the agent sees sufficiently many other foraging ants before the end of the observation period, it can switch at once to the self-grooming state.

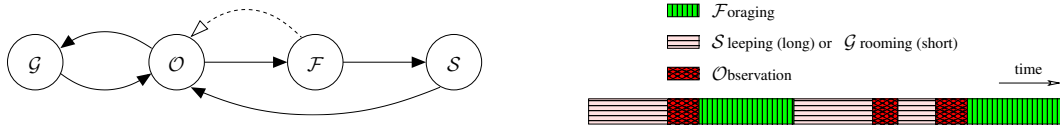

Figure 1: (Left) Transitions between the four agent states: $\mathcal{G}$rooming, $\mathcal{O}$bserving, $\mathcal{F}$oraging and $\mathcal{S}$leeping states. Black arrows denote transitions and the dotted arrow indicates an inhibitory message. (Right) An example of agent schedule.

The agent decisions only depend on the information exchanged between them, through agent neurons sending spikes to (respectively, receiving spikes from) other agents in the population. It must be emphasized that the proposed decision process does not assume the agent ability to "count" (here the number of its foraging neighbors). In the meanwhile, this process is deterministic, contrasting with the threshold-based probabilistic models used in [1, 2, 7].

## 2 The SpikeAnts spiking neuron network

This section describes the structure of the SpikeAnts model. Each ant agent is modelled by two spiking neurons. Any two agents $(i, j)$ are connected with an average density $\rho$ ($0 \leqslant \rho \leqslant 1$). The ant colony thus defines a sparsely connected network of spiking neurons, referred to as SNN.

### 2.1 Spiking neuron models

An agent is modelled by two coupled spiking neurons, respectively a Leaky Integrate-and-Fire (LIF) neuron [6, 14] and a Quadratic Integrate-and-Fire (QIF) neuron [8, 15]. These models of neuron are biologically plausible and they have been thoroughly studied. We shall show that their coupling achieves a frugal control of the agent behavior.

A LIF neuron fires a spike if its potential $V_p$ exceeds a threshold $\vartheta$. Upon firing a spike, $V_p$ is reset to $V_{\text{reset}}$. Formally:

$$\begin{cases} \frac{dV_p}{dt} = -\lambda(V_p(t) - V_{\text{rest}}) + I_{\text{exc}}(t), & \text{if } V_p < \vartheta \\ \text{else fires a spike and } V_p \text{ is set to } V_{\text{reset}}^p \end{cases}, \tag{1}$$

where $\lambda$ is the relaxation constant. $I_{\text{exc}}(t)$ models instantaneous synaptic interactions. Let **Pre** denote the set of presynaptic neurons (such that there exists an edge from every neuron in **Pre** and

the current neuron), and let $\mathbf{Train}_i$ denote the spike trains of the $i^{\text{th}}$ neuron in $\mathbf{Pre}$; then,

$$I_{\text{exc}}(t) = w \sum_{i \in \mathbf{Pre}} \sum_{j \in \mathbf{Train}_i} \delta(t - t_j^i), \tag{2}$$

where $w$ is a synaptic weight controlling the dynamics of the SNN (more in section 3.1), $\delta(.)$ is Dirac distribution and $t_j^i$ is the firing time of the $j^{\text{th}}$ spike from the $i^{\text{th}}$ presynaptic neuron.

The QIF neuron is described by the evolution of the potential $V_a$, compared to the resting potential $V_{\text{rest}}$ and an internal threshold $V_{\text{thres}}$. Additionally, it receives an internal signal $I_{\text{clock}}$ modelling a gap junction connection:

$$\begin{cases} \frac{dV_a}{dt} = \lambda(V_a(t) - V_{\text{rest}})(V_a(t) - V_{\text{thres}}) + I_{\text{inh}}(t) + I_{\text{clock}}(t), & \text{if } V_a < \vartheta \\ \text{else fires a spike and } V_a \text{ is set to } V_{\text{reset}}^a \end{cases} \tag{3}$$

Depending on whether the reset threshold is greater than the internal threshold ($V_{\text{reset}}^a \geqslant V_{\text{thres}}$), the QIF neuron is bistable [12], which motivated the choice of this neuron model. If $V_{\text{reset}}^a < V_{\text{thres}}$, the membrane potential $V_a$ stabilizes on $V_{\text{rest}}$ when there is no external perturbation, and the neuron thus exhibits an integrator behavior. When $V_{\text{reset}}^a \geqslant V_{\text{thres}}$, the neuron displays a bursting behavior and fires periodically.

## 2.2 The ant agent model

Each SpikeAnts agent mimics an ant. Its behavior is controlled after the competition between two coupled spiking neurons, an active one (QIF, Eq. (3)) and a passive one (LIF, Eq. (1)). The agent additionally involves an internal unit providing the $I_{\text{clock}}$ signal.

During the observation round, the ant makes its decision (whether it goes foraging) based on the competition between its active and passive neurons (Fig. 2). Both neurons are aware of the foraging neighbor ants. The signal emitted by these neighbors is an excitatory signal (respectively an inhibitory signal) for the passive (resp. active) neuron: $I_{\text{inh}}(t) = -I_{\text{exc}}(t)$. The active neuron additionally receives the excitatory signal $I_{\text{clock}}(t)$ of the internal clock unit.

In the case where the ant agent does not see too many foraging ants, the internal excitatory signal $I_{\text{clock}}(t)$ dominates the inhibitory signal $I_{\text{inh}}(t)$, the active neuron fires first and drives the ant to

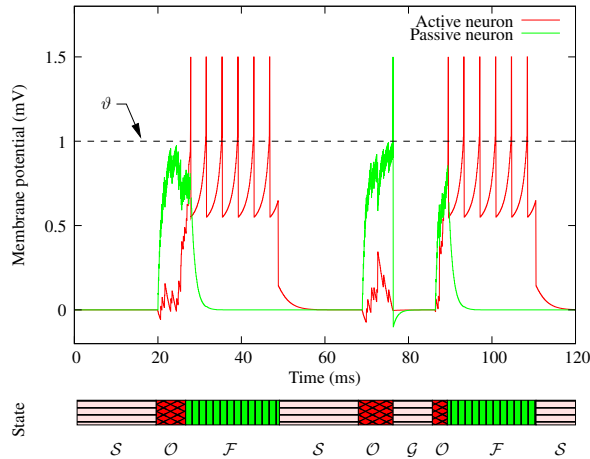

Figure 2: Membrane potentials of active (in dark/red) and passive (in grey/green) neurons. The dashed line indicates the threshold $\vartheta$. The first *observation* state starts at 20ms: the active neuron fires before the passive one, the agent thus goes foraging and the active neuron continues sending spikes during the whole foraging period (signalling its foraging behavior to other agents). After a *sleep* period (from circa 50 to 70ms), starts a second observation round. This time the passive neuron fires before the active one. The agent thus goes self-*grooming*, and switches to the observation state thereafter. During the last *observation* round, the active neuron wins again against the passive one, and the agent goes foraging.

foraging (first and last episode in Fig. 2). When foraging, the active neuron enters in a bursting phase and periodically sends a spike to the ant neighbors. Note that these spikes are only meaningful for the ants in observation state. After a foraging period (duration $t_{\mathcal{F}}$), the ant goes to sleep (duration $t_{\mathcal{S}}$). The sleeping state is triggered by a delayed connection between the internal unit and the active neuron.

Quite the contrary, if the ant sees many other foraging ants, the excitatory signal $I_{\mathrm{exc}}(t)$ drives the passive neuron to fire before the active one (second episode in Fig. 2), and the ant accordingly sets in a self-grooming state (duration $t_{\mathcal{G}}$). The decision making of the ant agent thus relies on the competition between its active and passive neurons. In particular, the number of spikes needed for an ant to go foraging or self-grooming depends on the temporal dynamics of the system; it varies from one observation episode to another. After some rest (self-grooming or sleeping states, with respective durations $t_{\mathcal{G}}$ and $t_{\mathcal{S}}$, $t_{\mathcal{G}} < t_{\mathcal{S}}$), the ant returns to the observation state.

As above-mentioned, incoming spikes are only relevant to the active and passive neurons of an observing ant. During the foraging and resting states, presynaptic spikes have no influence, which can be thought of as an intrinsic plasticity mechanism [21] driven by the internal unit. The internal unit can indeed be seen as the ant biological clock. In a further model, it will be replaced by a neural group interacting with active and passive neurons through intrinsic plasticity, e.g. using a transient increase of $\lambda$ for LIF and QIF neurons.

## 2.3 Model parameters

Overall, the SpikeAnts model is controlled by three types of parameters, respectively related to spiking neuron models, to ant agents (state durations) and to the whole population (size and connectivity of the SNN). The default parameter values used in the simulations are displayed in Table 1. The values of state durations are such that their ratio are not integers, in order to avoid spurious synchronizations. Note that state duration timescale is not significant at the ant colony level.

| Parameter type | Symbol | Description | Value | (units) |
|---|---|---|---|---|
| Neural | $\lambda$ | Membrane relaxation constant | 0.1 | $\mathrm{mV}^{-1}$ |
| | $V_{\mathrm{rest}}$ | Resting potential | 0.0 | mV |
| | $\vartheta$ | Spike firing threshold | 1.0 | mV |
| | $V_{\mathrm{reset}}^{p}$ | Passive neuron reset potential | -0.1 | mV |
| | $V_{\mathrm{thres}}$ | Active neuron bifurcation threshold | 0.5 | mV |
| | $V_{\mathrm{reset}}^{a}$ | Active neuron reset potential | 0.55 | mV |
| | $I_{\mathrm{clock}}$ | Active neuron constant input current | 0.1 | mV |
| | $w$ | Synaptic weight | 0.01 | $\mathrm{mV}^{-1}$ |
| Agent | $t_{\mathcal{F}}$ | Foraging duration | 47.1 | ms |
| | $t_{\mathcal{O}}$ | Maximum observation duration | 10.5 | ms |
| | $t_{\mathcal{S}}$ | Sleeping duration | 45.7 | ms |
| | $t_{\mathcal{G}}$ | Self-grooming duration | 16.7 | ms |
| Population | $\rho$ | Connection probability | 0.3 | |
| | $M$ | population size | 150 | agents |

Table 1: Neural, model and population parameters used in simulations.

# 3 Experiments

This section reports on the experimental study of the SpikeAnts model, first describing the experimental setting and the goals of experiments. The population behavior is measured after a global indicator, and the sensitivity thereof w.r.t. the SpikeAnts parameters is studied. Two compound control parameters, summarizing the model parameters and governing the emergent synchronization of the system are proposed. A consistent phase diagram depicting the global synchronization in the plane defined from both control parameters is displayed and discussed.

**Goals of experiments**    A first goal of experiments is to measure the global activity of the population, denoted $\mathcal{F}$ and defined as the overall time spent foraging:

$$\mathcal{F} = \sum_{t} n_{\mathcal{F}}(t) \tag{4}$$

where $n_{\mathcal{F}}(t)$ is the number of foraging agents at time $t$. The study focuses on the sensitivity of $\mathcal{F}$ w.r.t. the model parameters.

The second and most important goal of experiments is to study the temporal structure of the population activity. A synchronization indicator will be proposed and its sensitivity w.r.t. the model parameters will be examined.

**Experimental settings**  Each run starts with all ants initially sleeping. Each ant wakes up after some time uniformly drawn in $]0, 2t_S]$. Spiking neurons are simulated using a discrete time scheme: numerical simulations of the spiking neuron network are based on a clock-driven simulator, using Runge-Kutta method for the approximation of differential equations, with a small time step of 0.1ms to enforce numerical stability. Each run lasts for 100,000 time steps. All reported results are averaged over 10 independent runs.

## 3.1  Sensitivity analysis of the foraging effort

This section first examines how the overall foraging effort $\mathcal{F}$ depends on the size $M$ of the population, the connection rate $\rho$ and two neural parameters, the active neuron reset potential $V_{\text{reset}}^a$ and the synaptic weight $w$. The average $\bar{\mathcal{F}}$ is reported with its standard deviation in Fig. 3.

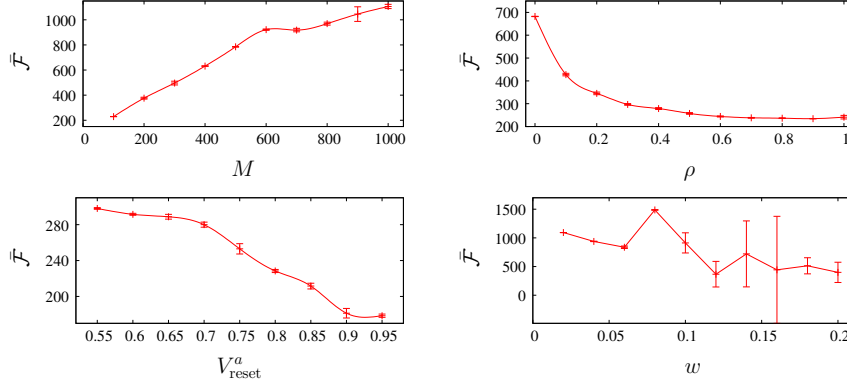

Figure 3: Sensitivity analysis of the average foraging effort $\bar{\mathcal{F}}$, versus population size $M$ (top left), connection probability $\rho$ (top right), active neuron reset potential $V_{\text{reset}}^a$ (bottom left) and synaptic weight $w$ (bottom right).

The overall foraging effort $\mathcal{F}$ was expected to linearly increase with the population size $M$. While it indeed increases with $M$, it displays a breaking down around $M$=600 (Fig. 3, top left); this unexpected change will be explained in section 3.2, and related to the increased variability of the population synchronization. $\mathcal{F}$ was expected to exponentially decrease with the connectivity $\rho$, and it does so (Fig. 3, top right): the more neighbors, the more likely an ant will see other foraging ants, and will thus avoid go foraging itself. Along the same line, $\mathcal{F}$ was expected to decrease with the reset potential $V_{\text{reset}}^a$: the closer $V_{\text{reset}}^a$ to $\vartheta$, the more spikes a foraging ant will sent, exciting other ants' passive neuron and thereby sending these ants to rest (Fig. 3, bottom left; the value of $\vartheta$ is 1, and $\mathcal{F}$ indeed goes to 0 as $V_{\text{reset}}^a$ goes to 1).

The most surprising result regards the influence of the synaptic weight $w$ (Fig. 3, bottom right). It was expected that high $w$ values would favor the triggering of passive neurons, and thus adversely affect the foraging effort. High $w$ values however mostly result in a high variance of $\mathcal{F}$. The interpretation proposed for this fact goes as follows. For low $w$ values, an ant behaves as a "good statistician", meaning that its decision is based on observing many other foraging agents. Accordingly, the foraging/resting ratio is very stable along time and across runs. As $w$ increases however, it makes it possible for an ant to take decisions based on few cues and the behavioral variability increases. More precisely, the $\mathcal{F}$ variance is low for small $w$ values (an ant makes its decision based on about 80 spikes for $w = 0.01$). The variance dramatically increases in a narrow region around $w = 0.15$; an ant makes its decision based on circa 6 spikes and small variations in the received spike trains might thus lead to different decisions, explaining the high variance of $\mathcal{F}$. For higher $w$

values however, the $\mathcal{F}$ variance decreases again. A close look at the experimental results reveals the existence of different temporal regimes with abrupt transitions among these, explaining the breaking down around $M = 600$ ants and the abrupt increase and decrease of $\mathcal{F}$ variance.

## 3.2 Emergent synchronization: Control parameters and phase transitions

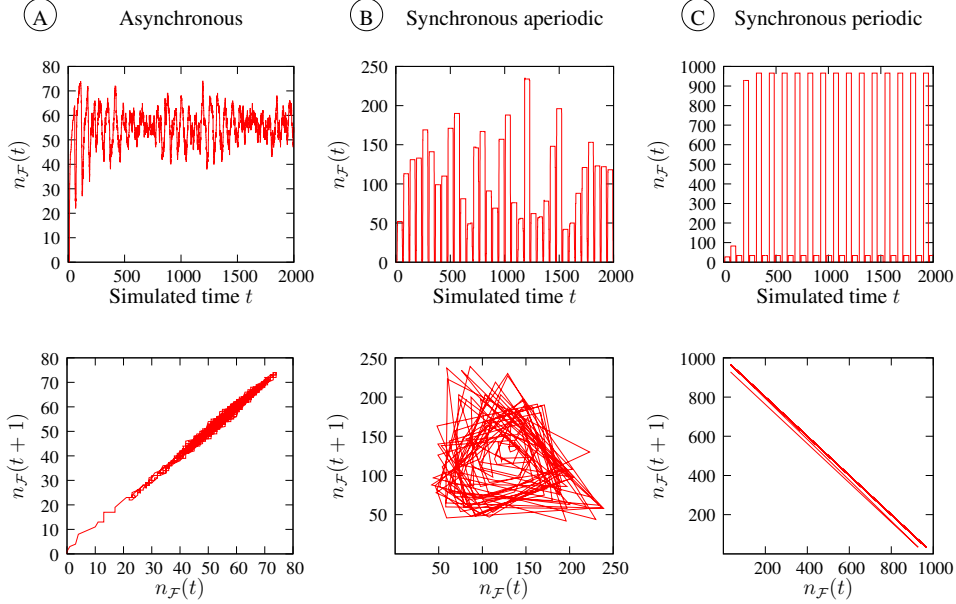

Figure 4: (Top row) Asynchronous, synchronous aperiodic and synchronous periodic patterns of activity (number of foraging ants versus time for $t = 1 \ldots 2,000$). (Bottom row) Temporal correlation of the activity for the above three patterns, for $t = 1 \ldots 100,000$.

The emergence of three synchronization patterns appears in the experimental results. The first one, referred to as *asynchronous* (Fig. 4, left), depicts a situation where each ant (almost) independently makes its own decisions. The second one, referred to as *synchronous* (Fig. 4, middle) displays some coordination among the ants; specifically, the number of foraging ants is piecewise constant, though varying from a time interval to another. The third pattern, referred to as *periodic synchronous* (Fig. 4, right) involves two stable subpopulations which forage alternatively; the population enters a bi-phase mode, as actually observed in some ants colonies [4, 5].

The difference between the three patterns of activity is most visible from the phase diagram plotting $n_{\mathcal{F}}(t + 1)$ vs $n_{\mathcal{F}}(t)$ (Fig. 4, bottom row; transient states are removed in the synchronized periodic and aperiodic regime for the sake of clarity). The orbit of the synchronous aperiodic activity indicates the presence of at least one attractor whereas the synchronous periodic activity displays a flip bifurcation.

The ergodicity of the SpikeAnts system is first analyzed based on the Lyapunov exponents, after the computation algorithms proposed in [22]. On asynchronous patterns, the mean value of the 5,000 Lyapunov exponents found with an 8 dimension analysis is $-0.01 \pm 0.1$. For synchronous aperiodic patterns, the mean value of the 3,500 Lyapunov exponents found with a 6 dimension analysis is also $-0.01 \pm 0.1$ (after discarding the transient states). Whereas the asynchronous and synchronous aperiodic activities lie at the edge of chaos, the periodic synchronous regime only displays large negative Lyapunov exponents, indicating a very stable behavior.

An entropy-based indicator is proposed to analyze the emergent synchronization of the SpikeAnts system. Let $\mathcal{I}$ denote the set of values $n_{\mathcal{F}}(t)$ (after pruning all transient time steps such that $n_{\mathcal{F}}(t) \neq n_{\mathcal{F}}(t + 1)$ and $n_{\mathcal{F}}(t) \neq n_{\mathcal{F}}(t - 1)$); the foraging histogram is defined by associating to each value $k$ in $\mathcal{I}$, the number $n_k$ of time steps such that $n_{\mathcal{F}}(t) = k$. The synchronization of the population is

finally measured from the histogram entropy $H$:

$$H = -\sum_{k \in \mathcal{I}} \frac{n_k}{\sum_m n_m} \log\left(\frac{n_k}{\sum_m n_m}\right) \tag{5}$$

The entropy of the asynchronous regime is zero, since all states are transient. The synchronous periodic regime, where two subpopulations alternatively forage, gets a low entropy ($< \log 2$). Finally, the synchronous aperiodic regime which involves a few dozens of subpopulations, gets a high entropy value. The transition from one regime to another one is clearly related to the model parameters. The goal thus becomes to identify the influential factors, best explaining the population behavior.

A first such influential factor, defined as $\rho\sqrt{M}$ and referred to as *sociability*, controls the amount of interactions between the ants. A high sociability enables the ants to base their foraging decision on reliable estimates of the current foraging activity, thus entailing a low variance of the global foraging effort.

A second influential factor, referred to as *receptivity*, is the ratio between the weight $w$ of the input signal and the subthreshold range (depending on the resting potential $V_{\text{rest}}$ and the spike firing threshold $\vartheta$). This ratio $\frac{w}{|\vartheta - V_{\text{rest}}|}$ indicates the amplitude of the depolarization induced by the input spike compared to the difference between rest and threshold. A high receptivity thus enables the ant to postpone its foraging decision based on few cues (i.e. visible foraging ants), thereby entailing a high variance of the global foraging effort.

The sociability and receptivity factors, referred to as *control parameters*, support a clear picture of the asynchronous, synchronous aperiodic and periodic synchronous patterns. The entropy (Fig. 5, left) and its variance (Fig. 5, right) are displayed in the 2D plane defined from the sociability and receptivity of the SpikeAnts system, defining the *phase diagram* of the SpikeAnts system.

For a low sociability and a high receptivity (region $A$ in Fig. 5), few interactions among ants take place and each ant makes its decisions based on few cues. In this region, the population is a collection of quasi independent individuals, and few ants (60 on average on Fig. 4) are foraging at any given time step.

For a higher sociability and a low receptivity (region $B$ in Fig. 5), ants see more of their peers and they base their decisions on reliable estimates of the foraging activity. A synchronization of the ant activities emerges, in the sense that many agents make their foraging decisions at the same time. Still, the synchronization remains aperiodic, i.e. the number of foraging ants varies from 50 to 240 (Fig. 4).

For a high sociability and a high receptivity (region $C$ in Fig. 5), ants see many of their peers and they make their decisions based on few cues. In this case a periodic synchronized regime is observed, where two subpopulations alternatively go foraging (the first one involves $\sim 950$ ants in Fig. 4).

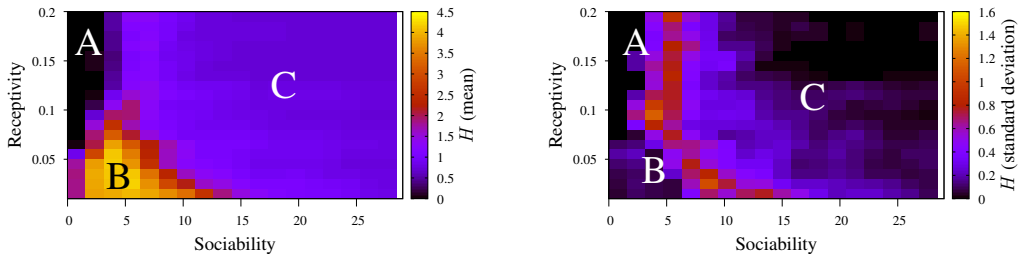

Figure 5: Emergence of synchronizations in the population activity: entropy $H$ (left) and variance of $H$ (right) versus the ant sociability and receptivity. The asynchronous pattern, with entropy $H = 0$ corresponds to a low sociability and high receptivity (region A). The synchronous aperiodic pattern, with high entropy, corresponds to a medium sociability and low receptivity (region B). The synchronous periodic pattern, $H \sim \log 2$, corresponds to both high sociability and receptivity (region C).

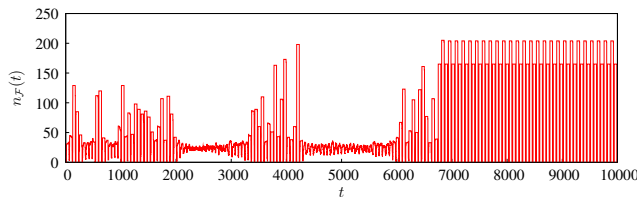

Figure 6: A representative simulation: the global behavior switches from a synchronous aperiodic regime to an asynchronous one before stabilizing in a periodic synchronous regime.

Complementary experiments show abrupt transitions between the different regimes in the borderline regions. Specifically, an asynchronous aperiodic regime (region $B$) is prone to evolve into an asynchronous (region $A$) or periodic synchronous (region $C$) regimes (Figure 6). Quite the contrary, the periodic synchronous regime is stable, i.e. the population does not get back to any other regime after the periodic synchronous regime is installed. The aperiodic synchronous regime, though less stable than the periodic one, is far more stable than the asynchronous one.

## 4  Discussion

The main contribution of this paper is a local and parsimonious model, accounting for individual decision making, which reproduces the emergence of synchronized activity in a complex system in a realistic way: the three different regimes obtained in simulation are comparable to the different patterns of activity observed in social insect colonies [7, 5, 4]. The synchronization patterns that emerge at the macroscopic scale can be fully controlled by several model parameters ruling the sociability of ants (whether an ant may observe many other ants) and their receptivity (whether an ant makes its foraging decision based on a few cues). The synchronization patterns are endogenous, with no external influence from the environment. Additionally, they do not rely on individual synchronizations, as each agent has a specific behavior, different from its neighbor and varying during simulation time.

To our best knowledge, the SpikeAnts model is the first one accounting for a population behavior and based on spiking neurons. SpikeAnts captures both spatial and temporal features of the complex system in a deterministic way (as opposed to stochastic models). It does not require any external constraints or data. Most importantly, it does not require the agent to feature sophisticated skills (e.g. "counting" its foraging neighbors). It is worth noting that SpikeAnts does not involve the resolution of differential equations: While spiking neurons are modelled in continuous time, their behavior is computed through finite differences, parameterized from the user-specified time step. In summary, SpikeAnts demonstrates that SNNs can be used to model a simple self-organizing system. It hopefully opens new perspectives for modelling emergent phenomena in complex systems.

A first perspective for further research is to investigate the temporal dynamics of spike trains using standard approaches from neuroscience. The underlying question is whether the population synchronization can be facilitated, e.g. in the transient regime, by making spiking neurons sensitive to the synchrony of spike trains. The role of inhibition and the role of the excitation/inhibition balance in the emergence of synchronized patterns will be studied. In particular, the impact on the phase diagram of individual parameter variations will be analyzed.

A second perspective is to endow SpikeAnts with some learning skills, e.g. adapting the connections weights $w$ with a local unsupervised learning rule (e.g. Spike-Timing-Dependent Plasticity), in order to optimize the collective efficiency of the population. Along the same line, the ability of SpikeAnts to cope with external perturbations (e.g. affecting the number of foraging ants) will be investigated.

### Acknowledgments

We thank Mathias Quoy, Université Cergy, for many fruitful discussions about complex systems, and helpful remarks about this paper. We thank Jean-Louis Deneubourg and José Halloy, Université Libre de Bruxelles, for many insights into the collective behavior of living systems. This work was supported by NSF grant No. PHY-9723972 and by the European Integrated Project SYMBRION.

# References

[1] E. Bonabeau, G. Theraulaz, and J.L. Deneubourg. Fixed response thresholds and the regulation of division of labor in insect societies. *Bulletin of Mathematical Biology*, 60(4):753–807–807, July 1998.

[2] E. Bonabeau, G. Theraulaz, and J.L. Deneubourg. The synchronization of recruitement-based activities in ants. *BioSystems*, 45:195–211, 1998.

[3] N. Brunel and X.J. Wang. What determines the frequency of fast network oscillations with irregular neural discharges? I. Synaptic dynamics and excitation-inhibition balance. *Journal of Neurophysiology*, 90(1):415–430, 2003.

[4] B.J. Cole. Short-term activity cycles in ants: Generation of periodicity by worker interaction. *The American Naturalist*, 137(2), 1991.

[5] N.R. Franks and S. Bryant. Rhythmical patterns of activity within the nest of ants. *Chemistry and Biology of Social Insects*, pages 122–123, 1987.

[6] W. Gerstner and W. Kistler. *Spiking Neuron Models: Single Neurons, Population, Plasticity*. Cambridge University Press, 2002.

[7] S. Goss and J.L. Deneubourg. Autocatalysis as a source of synchronised rythmical activity in social insects. *Insectes Sociaux*, 35(3):310–315, 1988.

[8] D. Hansel and G. Mato. Existence and stability of persistent states in large neuronal networks. *Physical Review Letters*, 86(18):4175–4178, April 2001.

[9] D.O. Hebb. *The Organization of Behaviour*. Wiley, New York, 1949.

[10] J.J. Hopfield and C.D. Brody. What is a moment? Transient synchrony as a collective mechanism for spatiotemporal integration. *Proc. Natl. Acad. Sci.*, 98(3):1282–1287, 2001.

[11] E.M. Izhikevich. Polychronization: Computation with spikes. *Neural Computation*, 18(2):245–282, 2006.

[12] E.M. Izhikevich. *Dynamical systems in neuroscience: the geometry of excitability and bursting*, chapter One-Dimensional Systems. MIT Press, 2007.

[13] H. Jaeger, W. Maass, and J. Principe. Special issue on echo state networks and liquid state machines (editorial). *Neural Networks*, 20(3):287–289, April 2007.

[14] B.W. Knight. Dynamics of encoding in a population of neurons. *The Journal of General Physiology*, 59(6):734–766, June 1972.

[15] P.E. Latham, B.J. Richmond, P.G. Nelson, and S. Nirenberg. Intrinsic dynamics in neuronal networks. i. theory. *Journal of Neurophysiology*, 83(2):808–827, February 2000.

[16] W. Liu, A.F.T. Winfield, J. Sa, J. Chen, and L. Dou. Towards energy optimisation: Emergent task allocation in a swarm of foraging robots. *Adaptive Behavior*, 15(3):289–305, 2007.

[17] R.E. Mirollo and S.H. Strogatz. Synchronization of pulse-coupled biological oscillators. *SIAM Journal on Applied Mathematics*, 50(6):1645–1662, 1990.

[18] H. Paugam-Moisy and S.M. Bohte. *Handbook of Natural Computing*, chapter 10. Computing with Spiking Neuron Networks. Springer, 2010. (in press).

[19] D. Phan, M.B. Gordon, and J.P. Nadal. *Cognitive Economics*, chapter Social interactions in economic theory: An insight from statistical mechanics, pages 335–358. Springer, 2004.

[20] B. Schrauwen, L. Büsing, and R. Legenstein. On computational power and the order-chaos phase transition in reservoir computing. In D. Koller, D. Schuurmans, Y. Bengio, and L. Bottou, editors, *Advances in Neural Information Processing Systems*, pages 1425–1432. MIT Press, 2008.

[21] J. Triesch. Synergies between intrinsic and synaptic plasticity mechanisms. *Neural Computation*, 19(4):885–909, 2007.

[22] A. Wolf, J. Swift, H. Swinney, and J. Vastano. Determining lyapunov exponents from a time series. *Physica D: Nonlinear Phenomena*, 16(3):285–317, 1985.

